# Nash Propagation for Loopy Graphical Games

**Luis E. Ortiz**
**Michael Kearns**
Department of Computer and Information Science
University of Pennsylvania
{leortiz,mkearns}@cis.upenn.edu

## Abstract

We introduce NashProp, an iterative and local message-passing algorithm for computing Nash equilibria in multi-player games represented by arbitrary undirected graphs. We provide a formal analysis and experimental evidence demonstrating that NashProp performs well on large graphical games with many loops, often converging in just a dozen iterations on graphs with hundreds of nodes.

NashProp generalizes the tree algorithm of (Kearns et al. 2001), and can be viewed as similar in spirit to belief propagation in probabilistic inference, and thus complements the recent work of (Vickrey and Koller 2002), who explored a junction tree approach. Thus, as for probabilistic inference, we have at least two promising general-purpose approaches to equilibria computation in graphs.

## 1 Introduction

There has been considerable recent interest in representational and algorithmic issues arising in multi-player game theory. One example is the recent work on *graphical games* (Kearns et al. 2001) (abbreviated KLS in the sequel). Here a multi-player game is represented by an undirected graph. The interpretation is that while the global equilibria of the game depend on the actions of all players, individual payoffs for a player are determined solely by his own action and the actions of his immediate neighbors in the graph. Like graphical models in probabilistic inference, graphical games may provide an exponentially more succinct representation than the standard "tabular" or normal form of the game. Also as for probabilistic inference, the problem of computing equilibria on arbitrary graphs is intractable in general, and so it is of interest to identify both natural special topologies permitting fast Nash computations, and good heuristics for general graphs.

KLS gave a dynamic programming algorithm for computing Nash equilibria in graphical games in which the underlying graph is a tree, and drew analogies to the polytree algorithm for probabilistic inference (Pearl 1988). A natural question following from this work is whether there are generalizations of the basic tree algorithm analogous to those for probabilistic inference. In probabilistic inference, there are two main approaches to generalizing the polytree algorithm. Roughly speaking, the first approach is to take an arbitrary graph and "turn it into a tree" via triangulation, and subsequently run the tree-based algorithm on the resulting *junction tree* (Lauritzen and Spiegelhalter 1988). This approach has the merit of being guaranteed to perform inference correctly, but the drawback of requiring the computation to be done on the junction tree. On highly loopy graphs, junction tree computations may require exponential time. The other broad approach is to simply run (an appropriate

generalization of) the polytree algorithm on the original loopy graph. This method garnered considerable interest when it was discovered that it sometimes performed quite well empirically, and was closely connected to the problem of decoding in Turbo Codes. Belief propagation has the merit of each iteration being quite efficient, but the drawback of having no guarantee of convergence in general (though recent theoretical work has established convergence for certain special cases (Weiss 2000)).

In recent work, (Vickrey and Koller 2002) proposed a number of heuristics for equilibria computation in graphical games, including a constraint satisfaction generalization of KLS that essentially provides a junction tree approach for arbitrary graphical games. They also gave promising experimental results for this heuristic on certain loopy graphs that result in manageable junction trees.

In this work, we introduce the NashProp algorithm, a different KLS generalization which provides an approach analogous to loopy belief propagation for graphical games. Like belief propagation, NashProp is a local message-passing algorithm that operates directly on the original graph of the game, requiring no triangulation or moralization [1] operations. NashProp is a two-phase algorithm. In the first phase, nodes exchange messages in the form of two-dimensional tables. The table player $U$ sends to neighboring player $V$ in the graph indicates the values $U$ "believes" he can play given a setting of $V$ and the information he has received in tables from his other neighbors, a kind of conditional Nash equilibrium. In the second phase of NashProp, the players attempt to incrementally construct an equilibrium obeying constraints imposed by the tables computed in the first phase.

Interestingly, we can provide rather strong theory for the first phase, proving that the tables must always converge, and result in a reduced search space that can never eliminate an equilibrium. When run using a discretization scheme introduced by KLS, the first phase of NashProp will actually converge in time polynomial in the size of the game representation.

We also report on a number of controlled experiments with NashProp on loopy graphs, including some that would be difficult via the junction tree approach due to the graph topology. The results appear to be quite encouraging, thus growing the body of heuristics available for computing equilibria in compactly represented games.

## 2 Preliminaries

The *normal* or *tabular* form of an $n$-player, two-action[2] game is defined by a set of $n$ matrices $M_i$ ($1 \leq i \leq n$), each with $n$ indices. The entry $M_i(\vec{x}) \in [0, 1]$ specifies the payoff to player $i$ when the joint action of the $n$ players is $\vec{x} \in \{0, 1\}^n$. Thus, each $M_i$ has $2^n$ entries. The actions 0 and 1 are the *pure strategies* of each player, while a *mixed* strategy for player $i$ is given by the probability $p_i \in [0, 1]$ that the player will play 0. For any joint mixed strategy, given by a product distribution $\vec{p}$, we define the expected payoff to player $i$ as $M_i(\vec{p}) = \mathbf{E}_{\vec{x} \sim \vec{p}}[M_i(\vec{x})]$, where $\vec{x} \sim \vec{p}$ indicates that each $x_j$ is 0 with probability $p_j$ and 1 with probability $1 - p_j$.

We use $\vec{p}[i : p_i']$ to denote the vector which is the same as $\vec{p}$ except in the $i$th component, where the value has been changed to $p_i'$. A *(Nash) equilibrium* for the game is a mixed strategy $\vec{p}$ such that for any player $i$, and for any $p_i' \in [0, 1]$, $M_i(\vec{p}) \geq M_i(\vec{p}[i : p_i'])$. (We say that $p_i$ is a *best response* to the rest of $\vec{p}$.) In other words, no player can improve their expected payoff by deviating unilaterally from a Nash equilibrium. The classic theorem of (Nash 1951) states that for any game, there exists a Nash equilibrium in the space of joint mixed strategies. We will also use a straightforward definition for approximate Nash equilibria. An *$\epsilon$-Nash equilibrium* is a mixed strategy $\vec{p}$ such that for any player $i$, and for any value $p_i' \in [0, 1]$, $M_i(\vec{p}) + \epsilon \geq M_i(\vec{p}[i : p_i'])$. (We say that $p_i$ is an *$\epsilon$-best response* to the rest of $\vec{p}$.) Thus, no player can improve their expected payoff by more than $\epsilon$ by

deviating unilaterally from an approximate Nash equilibrium.

The following definitions are due to KLS. An $n$-player *graphical game* is a pair $(G, \mathcal{M})$, where $G$ is an undirected graph on $n$ vertices and $\mathcal{M}$ is a set of $n$ matrices $M_i$ called the *local game matrices*. Each player is represented by a vertex in $G$, and the interpretation is that each player's payoff is determined solely by the actions in their local neighborhood in $G$. Thus the matrix $M_V \in \mathcal{M}$ has an index for each of the $k$ neighbors of $V$, and an index for $V$ itself, and for $\vec{x} \in [0,1]^{k+1}$, $M_V(\vec{x})$ denotes the payoff to $V$ when he and his $k$ neighbors play $\vec{x}$. The expected payoff under a mixed strategy $\vec{p} \in [0,1]^{k+1}$ is defined analogously. Note that in the two-action case, $M_V$ has $2^{k+1}$ entries, which may be considerably smaller than $2^n$.

Note that any game can be trivially represented as a graphical game by choosing $G$ to be the complete graph, and letting the local game matrices be the original tabular form matrices. However, any time in which the local neighborhoods in $G$ can be bounded by $k << n$, the graphical representation is exponentially smaller than the normal form. We are interested in heuristics that can exploit this succinctness computationally.

## 3   NashProp: Table-Passing Phase

The *table-passing* phase of NashProp proceeds in a series of rounds. In each round, every node will send a different binary-valued *table* to each of its neighbors in the graph. Thus, if vertices $V$ and $W$ are neighbors, the table sent from $V$ to $W$ in round $r$ shall be denoted $T^r_{WV}(w, v)$. Since the vertices are always clear from the lower-case table indices, we shall drop the subscript and simply write $T^r(w, v)$. This table is indexed by the continuum of possible mixed strategies $w, v \in [0, 1]$ for players $W$ and $V$, respectively. Intuitively, the binary value $T^r(w, v)$ indicates player $V$'s (possibly incorrect) "belief" that there exists a (global) Nash equilibrium in which $W = w$ and $V = v$.

As these tables are indexed by continuous values, it is not clear how they can be finitely represented. However, as in KLS, we shall shortly introduce a finite discretization of these tables whose resolution is dependent only on local neighborhood size, yet is sufficient to compute global (approximate) equilibria. For the sake of generality we shall work with the exact tables in the ensuing formal analysis, which will immediately apply to the approximation algorithm as well.

For every edge $(W, V)$, the table-passing phase initialization is $T^0(w, v) = 1$ for all $w, v \in [0, 1]$. Let us denote the neighbors of $V$ other than $W$ (if any) by $\vec{U} = (U_1, \ldots, U_{k-1})$. For each $w, v \in [0, 1]$, the table entry $T^{r+1}(w, v)$ is assigned the value 1 if and only if there exists a vector of mixed strategies $\vec{u} = (u_1, \ldots, u_{k-1}) \in [0, 1]^{k-1}$ for $\vec{U}$ such that

1.  $T^r(v, u_i) = 1$ for all $1 \leq i \leq k - 1$; and

2.  $V = v$ is a best response to $\vec{U} = \vec{u}, W = w$.

We shall call such a $\vec{u}$ a *witness* to $T^{r+1}(w, v) = 1$. If $V$ has no neighbors other than $W$, we define Condition 1 above to hold vacuously. If either condition is violated, we set $T^{r+1}(w, v) = 0$.

**Lemma 1** *For all edges $(W, V)$ and all $r > 0$, the table sent from $V$ to $W$ can only contract or remain the same:* $\{(w, v) : T^{r+1}(w, v) = 1\} \subseteq \{(w, v) : T^r(w, v) = 1\}$.

**Proof:**   By induction on $r$. The base case $r = 1$ holds trivially due to the table initialization to contain all 1 entries. For the induction, assume for contradiction that for some $r > 1$, there exists a pair of neighboring players $(W, V)$ and a strategy pair $(w, v) \in [0, 1]^2$ such that $T^r(w, v) = 0$ yet $T^{r+1}(w, v) = 1$. Since $T^{r+1}(w, v) = 1$, the definition of the table-passing phase implies that there exists a witness $\vec{u}$ for the neighbors $\vec{U}$ of $V$ other

than $W$ meeting Conditions 1 and 2 above. By induction, the fact that $T^r(v, u_i) = 1$ in Condition 1 implies that $T^{r-1}(v, u_i) = 1$ for all $i = 1, \ldots, k - 1$. Since $T^r(w, v) = 0$ it must be that $V = v$ is a not best response to $\vec{U} = \vec{u}, W = w$. But then $\vec{u}$ cannot be a witness to $T^{r+1}(w, v) = 1$, a contradiction. $\qquad\square$

Since all tables begin filled with 1 entries, and Lemma 1 states entries can only change from 1 to 0, the table-passing phase must converge:

**Theorem 2** *For all $(w, v) \in [0, 1]^2$, the limit $\lim_{r \to \infty} T^r(w, v) \equiv T^*(w, v)$ exists.*

It is also immediately obvious that the limit tables $\{T^*(w, v)\}$ must all simultaneously *balance* each other, in the sense of obeying Conditions 1 and 2. That is, we must have that for all edges $(W, V)$ and all $(w, v)$, $T^*(w, v) = 1$ implies the existence of a witness $\vec{u}$ for $\vec{U}$ such that $T^*(v, u_i) = 1$ for all $i$, and $V = v$ is a best response to $\vec{U} = \vec{u}, W = w$. If this were not true the tables would be altered by a single round of the table-passing phase.

We next establish that the table-passing phase will *never* eliminate any global Nash equilibria. Let $\vec{p} \in [0, 1]^n$ be any mixed strategy for the entire population of players, and let us use $\vec{p}[V]$ to denote the mixed strategy assigned to player $V$ by $\vec{p}$.

**Lemma 3** *Let $\vec{p} \in [0, 1]^n$ be a Nash equilibrium. Then for all rounds $r \geq 0$ of the table-passing phase, and every edge $(W, V)$, $T^r(\vec{p}[W], \vec{p}[V]) = 1$.*

**Proof:** By induction on $r$. The base case $r = 0$ holds trivially by the table initialization. By induction, for every $V$ and neighbor $U$ of $V$, $T^{r-1}(\vec{p}[V], \vec{p}[U]) = 1$, satisfying Condition 1 for $T^r(\vec{p}[W], \vec{p}[V]) = 1$. Condition 2 is immediately satisfied since $\vec{p}$ is a Nash equilibrium. $\qquad\square$

We can now establish a strong sense in which the set of balanced limit tables $\{T^*(w, v)\}$ characterizes the Nash equilibria of the global game. We say that $\vec{p}$ is *consistent* with the $\{T^*(w, v)\}$ if for every vertex $V$ with neighbors $W, \vec{U}$ we have $T^*(\vec{p}[W], \vec{p}[V]) = 1$, and $\vec{p}[\vec{U}]$ is a witness to this value. In other words, every edge assignment made in $\vec{p}$ is "allowed" by the $\{T^*(w, v)\}$, and furthermore the neighborhood assignments made by $\vec{p}$ are witnesses.

**Theorem 4** *Let $\vec{p} \in [0, 1]^n$ be any global mixed strategy. Then $\vec{p}$ is consistent with the balanced limit tables $\{T^*(w, v)\}$ if and only if it is a Nash equilibrium.*

**Proof:** The forward direction is easy. If $\vec{p}$ is consistent with the $\{T^*(w, v)\}$, then by definition, for all $V$, $V = \vec{p}[V]$ is a best response to the local neighborhood $W = \vec{p}[W], \vec{U} = \vec{p}[\vec{U}]$. Hence, $\vec{p}$ is a Nash equilibrium.

For the other direction, if $\vec{p}$ is a Nash equilibrium, then for all $V$, $V = \vec{p}[V]$ is certainly a best response to the strategy of its neighbors $W = \vec{p}[W], \vec{U} = \vec{p}[\vec{U}]$. So for consistency with the $\{T^*(w, v)\}$, it remains to show that for every player $V$ and its neighbors $W, \vec{U}$, $T^*(\vec{p}[V], \vec{p}[W]) = 1$ and $T^*(\vec{p}[V], \vec{p}[U_i]) = 1$ for all $i$. This has already been established in Lemma 3. $\qquad\square$

Theorem 4 is important because it establishes that the table-passing phase provides us with an alternative — and hopefully vastly reduced — seach space for Nash equilibria. Rather than search for equilibria in the space of all mixed strategies, Theorem 4 asserts that we can limit our search to the space of $\vec{p}$ that are consistent with the balanced limit tables $\{T^*(w, v)\}$, with no fear of missing equilibria. The demand for consistency with the limit tables is a locally *stronger* demand than merely asking for a player to be playing a best response to its neighborhood. Heuristics for searching this constrained space are the topic of Section 5.

But first let us ask in what ways the search space defined by the $\{T^*(w,v)\}$ might constitute a significant reduction. The most obvious case is that in which many of the tables contain a large fraction of 0 entries, since every such entry eliminates all mixed strategies in which the corresponding pair of vertices plays the corresponding pair of values. As we shall see in the discussion of experimental results, such behavior seems to occur in many — but certainly not all — interesting cases. We shall also see that even when such reduction does not occur, the underlying graphical structure of the game may still yield significant computational benefits in the search for a consistent mixed strategy.

## 4 Approximate Tables

Thus far we have assumed that the binary-valued tables $T^r(w,v)$ have continuous indices $w$ and $v$, and thus it is not clear how they can be finitely represented [3]. Here we briefly address this issue by asserting that it can be handled using the discretization scheme of KLS. More precisely, in that work it was established that if we restrict all table indices to only assume discrete values that are multiples of $\tau$, and we relax Condition 2 in the definition of the table-passing phase to ask that $V = v$ be only an $\epsilon$-best response to $W = w, \vec{U} = u$, then the choice $\tau = \epsilon/(2^{k+1}k\log(k))$ suffices to preserve $\epsilon$-Nash equilibria in the tables. Here $k$ is the maximum degree of any node in the graph. The total number of entries in each table will be $(1/\tau)^2$ and thus exponential in $k$, but the payoff matrices for the players are already exponential in $k$, so our tables remain polynomial in the size of the graphical game representation. The crucial point established in KLS is that the required resolution is *independent* of the total number of players. It is easily verified that none of the key results establishing this fact (specifically, Lemmas 2, 3 and 4 of KLS) depend on the underlying graph being a tree, but hold for all graphical games.

Precise analogues of all the results of the preceding section can thus be established for the discretized instantiation of the table-passing phase (details omitted). In particular, the table-passing phase will now converge to *finite* balanced limit tables, and consistency with these tables characterizes $\epsilon$-Nash equilibria. Furthermore, since every round prior to convergence must change at least one entry in one table, the table-passing phase must thus converge in at most $nk/\tau^2$ rounds, which is again polynomial in the size of the game representation. Each round of the table-passing phase takes at most on the order of $nk/\tau^{k+1}$ computational steps in the worst case (though possibly considerably less), giving a total running time to the table-passing phase that scales polynomially with the size of the game.

We note that the discretization of each player's space of mixed strategies allows one to formulate the problem of computing an approximate NE in a graphical game as a CSP(Vickrey and Koller 2002), and there is a precise connection between NashProp and constraint propagation algorithms for (generalized) arc consistency in constraint networks [4].

## 5 NashProp: Assignment-Passing Phase

We have already suggested that the tables $\{T^*(w,v)\}$ represent a solution space that may be considerably smaller than the set of all mixed strategies. We now describe heuristics for searching this space for a Nash equilibrium. For this it will be convenient to define, for each vertex $V$, its *projection set* $P^*(v)$, which is indexed by the possible values $v \in [0,1]$ (or by their allowed values in the aforementioned discretization scheme). The purpose of $P^*(v)$ is simply to consolidate the information sent to $V$ by all of its neighbors. Thus, if $\vec{U}$ are all the neighbors of $V$, we define $P^*(v)$ to be 1 if and only if there exists $\vec{u}$ (again called a *witness* to $P^*(v) = 1$) such that $T(v, u_i) = 1$ for all $i$, and $V = v$ is a best response to $\vec{U} = \vec{u}$; otherwise we define $P^*(v)$ to be 0.

If $\vec{p}$ is any global mixed strategy, it is easily verified that $\vec{p}$ is consistent with the $\{T^*(w,v)\}$

if and only if $P^*(\vec{p}[V]) = 1$ for all nodes $V$, with the assignment of the neighbors of $V$ in $\vec{p}$ as a witness. The first step of the assignment-passing phase of NashProp is thus the computation of the $P^*(v)$ at each vertex $V$, which is again a local computation in the graph. Neighboring nodes $V$ and $W$ also exchange their projections $P^*(v)$ and $P^*(w)$.

Let us begin by noting that the search space for a Nash equilibrium is immediately reduced to the cross-product of the projection sets by Theorem 4, so if the table-passing phase has resulted in many 0 values in the projections, even an exhaustive search across this (discretized) cross-product space may sometimes quickly yield a solution. However, we would obviously prefer a solution that exploits the local topology of the solution space given by the graph. At a high level, such a local search algorithm is straightforward:

1. Initialization: Choose any node $V$ and any values $v, \vec{u}$ such that $P^*(v) = 1$ with witness $\vec{u}$, and $P^*(u_i) = 1$ for all $i$. $V$ assigns itself value $v$, and assigns each of its neighbors $U_i$ the value $u_i$.

2. Pick the next node $V$ (in some fixed ordering) that has already been assigned some value $v$. If there is a partial assignment to the neighbors of $V$, attempt to extend it to a witness $\vec{u}$ to $P(v) = 1$ such that $P^*(u_i) = 1$ for all $i$, and assign any previously unassigned neighbors their values in this witness. If all the neighbors of $V$ have been assigned, make sure $V = v$ is a best response.

Thus, the first vertex chosen assigns both itself and all of its neighbors, but afterwards vertices assign only (some of) their neighbors, and receive their own values from a neighbor. It is easily verified that if this process succeeds in assigning all vertices, the resulting mixed strategy is consistent with the $\{T^*(w, v)\}$ and thus a Nash equilibrium (or approximate equilibrium in the discretized case). The difficulty, of course, is that the inductive step of the assignment-passing phase may fail due to cycles in the graph — we may reach a node $V$ whose neighbor partial assignment cannot be extended, or whose assigned value $V = v$ is not a best response to its complete neighborhood assignment. In this case, as with any structured local search phase, we have reached a failure point and must backtrack.

The overall NashProp algorithm thus consists of the (always converging) table-passing phase followed by the backtracking local assignment-passing phase. NashProp directly generalizes the algorithm of KLS, and as such, on certain special topologies such as trees may provably yield efficient computation of equilibria. Here we have shown that NashProp enjoys several natural and desirable properties even on arbitrary graphs. We now turn to some experimental investigation of NashProp on graphs containing cycles.

## 6   Experimental Results

We have implemented the NashProp algorithm (with distinct table-passing and assignment-passing [5] phases) as described, and run a series of controlled experiments on loopy graphs of varying size and topology. As discussed in Section 4, there is a relationship suggested by the KLS analysis between the table resolution $\tau$ and the global approximation quality $\epsilon$, but in practice this relationship may be pessimistic (Vickrey and Koller 2002) . Our implementation thus takes both $\tau$ and $\epsilon$ as inputs, and attempts to find an $\epsilon$-Nash equilibrium running NashProp on tables of resolution $\tau$.

We first draw attention to Figure 1, in which we provide a visual display of the evolution of the tables computed by the NashProp table-passing phase for a small (3 by 3) grid game. Note that for this game, the table-passing phase constrains the search space tremendously — so much so that the projection sets entirely determine the unique equilibrium, and the assignment-passing phase is superfluous. This is of course ideal behavior.

The main results of our controlled experiments are summarized in Figure 2. One of our

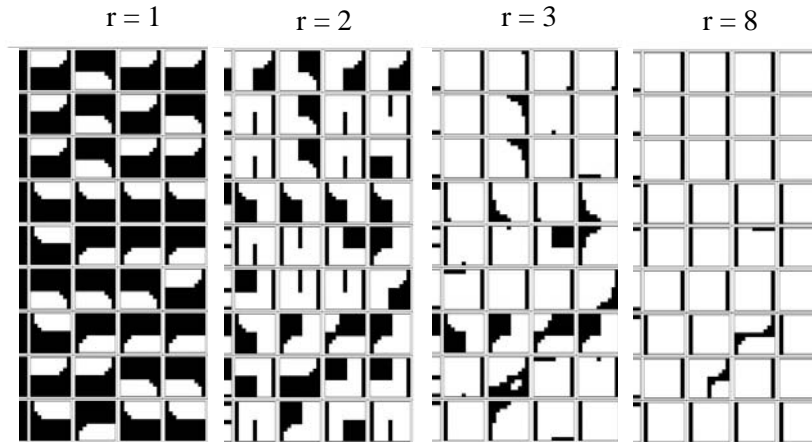

Figure 1: Visual display of the NashProp table-passing phase after rounds 1,2 and 3 and 8 (where convergence occurs). Each row shows first the projection set, then the four outbound tables, for each of the 9 players in a 3 by 3 grid. For the reward functions, each player has a distinct preference for one of his two actions. For 15 of the 16 possible settings of his 4 neighbors, this preference is the same, but for the remaining setting it is reversed. It is easily verified that every player's payoff depends on all of his neighbors. (Settings used: $\tau = 0.1$, $\epsilon = 0.05$).

primary interests is how the number of rounds in each of the two phases — and therefore the overall running time — scales with the size and complexity of the graph. More detail is provided in the caption, but we created graphs varying in size from 5 to 100 nodes with a number of different topologies: single cycles; single cycles to which a varying number of chords were added, which generates considerably more cycles in the graph; grids; and "ring of rings" (Vickrey and Koller 2002). We also experimented with local payoff matrices in which each entry was chosen randomly from $[0, 1]$, and with "biased" rewards, in which for some $\ell$ fixed number of the settings of its neighbors, each node has a strong preference for one of their actions, and in the remaining settings, a strong preference for the other. The $\ell$ settings were chosen randomly subject to the constraint that no neighbor is marginalized (thus no simplification of the graph is possible). These classes of graphs seems to generate a nice variability in the relative speed of the table-passing and assignment-passing phases of NashProp, which is why we chose them.

We now make a number of remarks regarding the NashProp experiments. First, and most basically, these preliminary results indicate that the algorithm performs well across a range of loopy topologies, including some (such as grids and cycles with many chords) that might pose computational challenges for junction tree approaches as the number of players becomes large. Excluding the small fraction of trials in which the assignment-passing phase failed to find a solution, even on grid and loopy chord graphs with 100 nodes, we find convergence of both the table and assignment-passing phases in less than a dozen rounds.

We next note that there is considerable variation across topologies (and little within) in the amount of work done by the table-passing phase, both in terms of the expected number of rounds to convergence, and the fraction of 0 entries that have been computed at completion. For example, for cycles the amount of work in both senses is at its highest, while for grids with random rewards it is lowest. For grids and chordal cycles, decreasing the value of $\ell$ (and thus increasing the bias of the payoff matrices) generally causes more to be accomplished by the table-passing phase. Intuitively, when rewards are entirely random and unbiased, nodes with large degrees will tend to rarely or never compute 0s in their

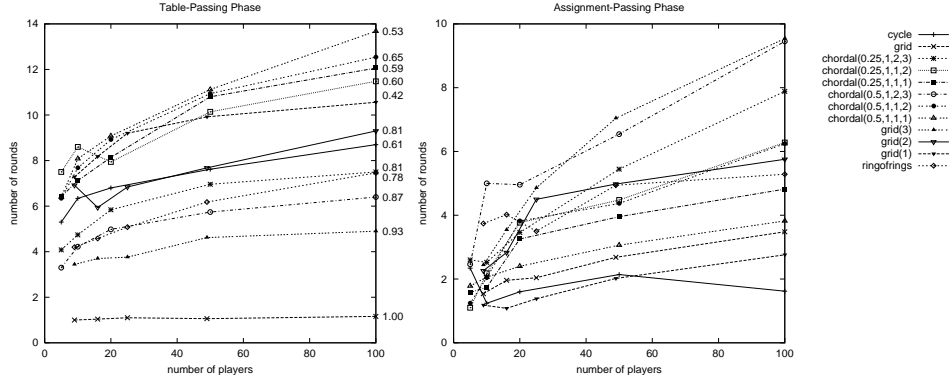

Figure 2: Plots showing the number of rounds taken by the NashProp table-passing (left) and assignment-passing (right) phases in computing an equilibrium, for a variety of different graph topologies. The $x$-axis shows the total number of vertices in the graph. Topologies and rewards examined included cycles, grids and 'ring of rings'(Vickrey and Koller 2002) with random rewards (denoted *cycle*, *grid* and *ringofrings* in the legend); cycles with a fraction $c$ of random chords added, and with biased rewards in which nodes of degree 2 have $\ell = \ell_2$, degree 3 have $\ell = \ell_3$, and degree 4 have $\ell = \ell_4$ (see text for definition of $\ell$), denoted *chordal*$(c, \ell_2, \ell_3, \ell_4)$; and grids with biased rewards with $\ell$, denoted *grid*$(\ell)$. Each data point represents averages over 50 trials for the given topology and number of vertices. In the table-passing plot, each curve is also annotated with the average fraction of 1 values in the converged tables. For cycles, settings used were $\tau = 0.1, \epsilon = 0.05$; for ring of rings, $\tau = 0.2, \epsilon = 0.06$; for all other classes, $\tau = 0.2, \epsilon = 0.2$.

outbound tables — there have too many neighbors whose combined setting can act as a witnesses for a 1 in an outbound table.

However, as suggested by the theory, greater progress (and computation) in the table-passing phase pays dividends in the assignment-passing phase, since the search space may have been dramatically reduced. For example, for chordal and grid graphs with biased rewards, the ordering of plots by convergence time is essentially reversed from the table-passing to assignment-passing phases. This suggests that, when it occurs, the additional convergence time in the table-passing phase is worth the investment. However, we again note that even for the least useful table-passing phase (for grids with random rewards), the assignment-passing phase (which thus exploits the graph structure alone) still manages to find an equilibrium rapidly.

## Footnotes

[1] Unlike for inference, moralization may be required for games even on undirected graphs.

[2] For simplicity, we describe our results for two actions, but they generalize to multi-action games.

[3]We note that the KLS proof that the exact tables must admit a rectilinear representation holds generally, but we cannot bound their complexity here.

[4]We are grateful to Michael Littman for helping us establish this connection.

[5]We did not implement backtracking, but this caused an overall rate of failure of only 3% across all 3000 runs described here.

## References

M. Kearns, M. Littman, and S. Singh. Graphical models for game theory. In *Proceedings of the Conference on Uncertainty in Artificial Intelligence*, pages 253–260, 2001.

S. Lauritzen and D. Spiegelhalter. Local computations with probabilities on graphical structures and their application to expert systems. *J. Royal Stat. Soc. B*, 50(2):157–224, 1988.

J. F. Nash. Non-cooperative games. *Annals of Mathematics*, 54:286–295, 1951.

J. Pearl. *Probabilistic Reasoning in Intelligent Systems*. Morgan Kaufmann, 1988.

D. Vickrey and D. Koller. Multi-agent algorithms for solving graphical games. In *Proceedings of the National Conference on Artificial Intelligence (AAAI)*, 2002. To appear.

Yair Weiss. Correctness of local probability propagation in graphical models with loops. *Neural Computation*, 12(1):1–41, 2000.
